# Movement extraction by detecting dynamics switches and repetitions

**Silvia Chiappa**
Statistical Laboratory
Wilberforce Road, Cambridge, UK
silvia@statslab.cam.ac.uk

**Jan Peters**
Max Planck Institute for Biological Cybernetics
Spemannstrasse 38, Tuebingen, Germany
jan.peters@tuebingen.mpg.de

## Abstract

Many time-series such as human movement data consist of a sequence of basic actions, e.g., forehands and backhands in tennis. Automatically extracting and characterizing such actions is an important problem for a variety of different applications. In this paper, we present a probabilistic segmentation approach in which an observed time-series is modeled as a concatenation of segments corresponding to different basic actions. Each segment is generated through a noisy transformation of one of a few hidden trajectories representing different types of movement, with possible time re-scaling. We analyze three different approximation methods for dealing with model intractability, and demonstrate how the proposed approach can successfully segment table tennis movements recorded using a robot arm as haptic input device.

## 1  Introduction

Motion capture systems have become widespread in many application areas such as robotics [18], physical therapy, sports sciences [10], virtual reality [15], artificial movie generation [13], computer games [1], etc. These systems are used for extracting the movement templates characterizing basic actions contained in their recordings. In physical therapy and sports sciences, these templates are employed to analyze patient's progress or sports professional's movements; in robotics, virtual reality, movie generation or computer games, they become the basic elements for composing complex actions.

In order to obtain the movement templates, boundaries between actions need to be detected. Furthermore, fundamental similarities and differences in the dynamics underlying different actions need to be captured. For example, in a recording from a game of table tennis, observations corresponding to different actions can differ, due to different goals for hitting the ball, racket speeds, desired ball interaction, etc. The system needs to determine whether this dissimilarity corresponds to substantially diverse types of underlying movements (such as in the case of a forehand and a backhand), or not (such as in the case of two forehands that differ only in speed).

To date, most approaches addressed the problem by using considerable manual interaction [16]; an important advancement would be to develop an automatic method that requires little human intervention. In this paper, we present a probabilistic model in which actions are assumed to arise from noisy transformations of a small set of hidden trajectories, each representing a different movement template, with non-linear time re-scaling accounting for differences in action durations. Action boundaries are explicitly modeled through a set of discrete random variables. Segmentation is obtained by inferring, at each time-step, the position of the observations in the current action and the underlying movement template. To guide segmentation, we impose constraints on the minimum and maximum duration that each action can have.

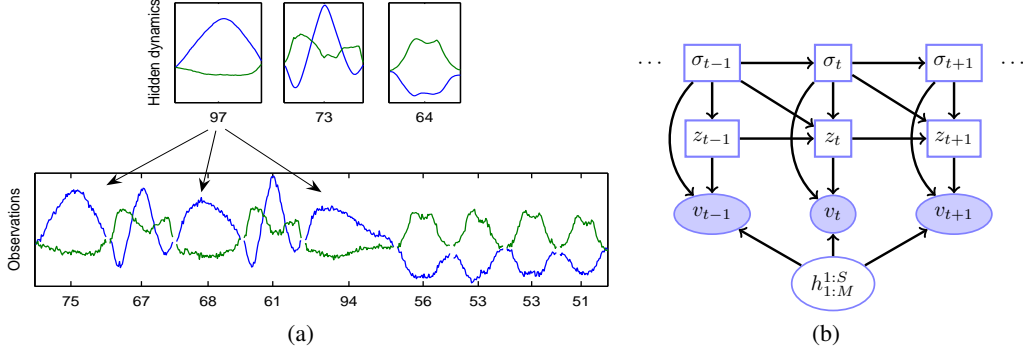

Figure 1: (a) The hidden dynamics shown on the top layer are assumed to generate the time-series at the bottom. (b) Belief network representation of the proposed segmentation model. Rectangular nodes indicate discrete variables, while (filled) oval nodes indicate (observed) continuous variables.

We apply the model to a human game of table tennis recorded with a Barrett WAM used as a haptic input device, and show that we can obtain a meaningful segmentation of the time-series.

## 2   The Segmentation Model

In the proposed segmentation approach, the observations originate from a set of continuous-valued hidden trajectories, each representing a different movement template. Specifically, we assume that the observed time-series consists of a concatenation of segments (basic actions), each generated through a noisy transformation of one of the hidden trajectories, with possible time re-scaling. This generation process is illustrated in Figure 1 (a), where the observations on the lower graph are generated from the three underlying hidden trajectories on the upper graph. Time re-scaling happens during the generation process, e.g., the first hidden trajectory of length 97 gives rise to three segments of length 75, 68 and 94 respectively.

The observed time-series and the $S$ hidden trajectories are represented by the continuous random variables[1] $v_{1:T} \equiv v_1, \ldots, v_T$ ($v_t \in \Re^V$) and $h_{1:M}^{1:S} \equiv h_{1:M}^1, \ldots, h_{1:M}^S$ ($h_m^i \in \Re^H$), respectively. Furthermore, we introduce two sets of discrete random variables $\sigma_{1:T}$ and $z_{1:T}$. The first set is used to infer which movement template generated the observations at each time-step, to detect action boundaries, and to define hard constraints on the minimum and maximum duration of each observed action. The second set is used to model time re-scaling from the hidden trajectories to the observations. We assume that the joint distribution of these variables factorizes as follows

$$p(h_{1:M}^{1:S})\prod_t p(v_t|h_{1:M}^{1:S}, z_t, \sigma_t)p(z_t|z_{t-1}, \sigma_{t-1:t})p(\sigma_t|\sigma_{t-1}).$$

These independence relations are graphically represented by the belief network of Figure 1 (b).

The variable $\sigma_t$ is a triple $\sigma_t = \{s_t, d_t, c_t\}$ with a similar role as in regime-switching models with explicit regime-duration distribution (ERDMs) [4]. The variable $s_t \in \{1, \ldots, S\}$ indicates which of the $S$ hidden trajectories underlies the observations at time $t$. The duration variables $d_t$ specifies the time interval spanned by the observations forming the current action, and takes a value between $d_{\min}$ and $d_{\max}$. The count variable $c_t$ indicates the time distance to the beginning of the next action, taking value $c_t = d_t$ and $c_t = 1$ respectively at the beginning and end of an action. More specifically, we define $p(\sigma_t|\sigma_{t-1}) = p(c_t|d_t, c_{t-1})p(d_t|d_{t-1}, c_{t-1})p(s_t|s_{t-1}, c_{t-1})$ with[2]

$$p(s_t|s_{t-1}, c_{t-1}) = \begin{cases} \pi_{s_t, s_{t-1}} & \text{if } c_{t-1}=1, \\ \delta(s_t=s_{t-1}) & \text{if } c_{t-1}>1, \end{cases}$$

$$p(d_t|d_{t-1}, c_{t-1}) = \begin{cases} \rho_{d_t} & \text{if } c_{t-1}=1, \\ \delta(d_t=d_{t-1}) & \text{if } c_{t-1}>1, \end{cases}$$

$$p(c_t|d_t, c_{t-1}) = \begin{cases} \delta(c_t=d_t) & \text{if } c_{t-1}=1, \\ \delta(c_t=c_{t-1}-1) & \text{if } c_{t-1}>1, \end{cases}$$

where $\delta(x=y)=1$ if $x=y$ and $\delta(x=y)=0$ if $x \neq y$, $\pi$ is a matrix specifying the time-invariant dynamics-switching distribution, and $\rho$ is a vector defining the action-duration distribution.

The variable $z_t$ indicates which of the $M$ elements in the hidden trajectory generated the observations $v_t$. We define $p(z_t|z_{t-1}, \sigma_{t-1:t}) = p(z_t|z_{t-1}, d_t, c_{t-1:t})$ with

$$p(z_t|z_{t-1}, d_t, c_{t-1:t}) = \begin{cases} \tilde{\psi}_{z_t}^{d_t, c_t} & \text{if } c_{t-1}=1, \\ \psi_{z_t, z_{t-1}}^{d_t, c_t} & \text{if } c_{t-1}>1. \end{cases}$$

The vector $\tilde{\psi}^{d_t, c_t}$ and the matrix $\psi^{d_t, c_t}$ encode two constraints[3]. First, $z_t - z_{t-1} \in \{1, \ldots, w_{\max}\}$ ensures that subsequent observations are generated by subsequent elements of the hidden trajectory and imposes a limit on the magnitude of time-warping. Second, $z_t \in \{d_t - c_t + 1, \ldots, M - c_t + 1\}$ accounts for the $d_t - c_t$ and $c_t - 1$ observations preceding and following $v_t$ in the action.

The hidden trajectories follow independent linear Markovian dynamics with Gaussian noise, that is

$$p(h_m^i|h_{m-1}^i) = \mathcal{N}(F^i h_{m-1}^i, \Sigma_H^i), h_1^i \sim \mathcal{N}(\mu^i, \Sigma^i).$$

Finally, the observations are generated from a linear transformation of the hidden variables with Gaussian noise

$$p(v_t|h_{1:M}^{1:S}, z_t, \sigma_t) = \mathcal{N}(G^{s_t} h_{z_t}^{s_t} + \lambda_{d_t, t+c_t-1}, \Sigma_V^{s_t}),$$

where the term $\lambda_{d_t, t+c_t-1}$ is common to all observations belonging to the same action and allows for spatial translation.

The generative process underlying the model is described in detail in[4] Table 1.

The set $\Theta$ of unknown model parameters is given by $\{F^{1:S}, G^{1:S}, \Sigma_H^{1:S}, \Sigma_V^{1:S}, \mu^{1:S}, \Sigma^{1:S}, \pi, \tilde{\pi}, \rho, \psi, \tilde{\psi}, \lambda\}$. After learning $\Theta$, we can sample a segmentation from $p(\sigma_{1:T}|v_{1:T})$ or compute the most likely segmentation $\sigma_{1:T}^* = \arg\max_{\sigma_{1:T}} p(\sigma_{1:T}|v_{1:T})$[5].

Table 1: Model's Generation Mechanism

> **for** $i = 1, \ldots, S$ **do**
>   generate hidden trajectory $i$
>   $h_1^i \sim \mathcal{N}(\mu^i, \Sigma^i)$
>   $h_m^i = F^i h_{m-1}^i + \eta_m^h, \eta_m^h \sim \mathcal{N}(0, \Sigma_H^i)$
> set $t = 1$
> **for action** $a = 1, \ldots, A$ **do**
>   sample a dynamics type $s_t \sim \pi_{:, s_{t-1}}$
>   sample a duration $d_t \sim \rho$
>   mark the beginning of the action $c_t = d_t$
>   **while** $c_t \geq 1$ **do**
>     sample time-warping $z_t \sim \psi_{:, z_{t-1}}^{d_t, c_t}$
>     generate the observations
>     $v_t = G^{s_t} h_{z_t}^{s_t} + \lambda_{d_t, t+c_t-1} + \eta_t^v$
>     $\eta_t^v \sim \mathcal{N}(0, \Sigma_V^{s_t})$
>     $t = t + 1$
>     **if** $c_{t-1} > 1$ **do**
>       $s_t = s_{t-1}, d_t = d_{t-1}, c_t = c_{t-1} - 1$

**Relation to previous models.** From a modeling point of view, the presented method builds on previous approaches that consider the observed time-series as time-warped transformations of one or several continuous-valued hidden trajectories. In [11], the authors introduced a model in which different time-series are assumed to be generated by a single continuous-valued latent trace, with spatial and time re-scaling. This model was used to align speech sequences. In [6], a modified version of such a model was employed in the domain of helicopter flight to learn a desired trajectory from demonstrations. In [12] and [14], the authors considered the case in which each time-series is generated by one of a set of different hidden trajectories. None of these models can deal with the situation in which possibly different dynamics underlie different segments of the same time-series.

From an application point of view, previous segmentation systems for extracting basic movements employed considerable human intervention [16]. On the other hand, automatic probabilistic methods for modeling movement templates assumed that the time-series data was pre-segmented into basic movements [5, 17].

# 3 Inference and Learning

The interaction between the continuous and discrete hidden variables renders the computation of the posterior distributions required for learning and sampling a segmentation intractable. In this section, we present and analyze three different approximation methods for dealing with this problem. In the first (variational) method, $p(h_{1:M}^{1:S}, z_{1:T}, \sigma_{1:T}|v_{1:T}, \Theta)$ is approximated with a simpler distribution $q$, and the optimal $q$ and $\Theta$ are found by maximizing a tractable lower bound on the log-likelihood using an Expectation-Maximization (EM) approach. In the second (maximum a posteriori) method, we estimate the most likely set of hidden trajectories and $\Theta$ by maximizing $p(h_{1:M}^{1:S}, v_{1:T}|\Theta)$ using an EM approach. In the third (Gibbs sampling) method, we use stochastic EM [3] with Gibbs sampling.

## 3.1 Variational Method

In the variational approximation, we introduce a distribution $q$ in which the problematic dependence between the hidden dynamics and the segmentation and time-warping variables is relaxed, that is[6]

$$q(h_{1:M}^{1:S}, z_{1:T}, \sigma_{1:T}) = q(h_{1:M}^{1:S})q(z_{1:T}|\sigma_{1:T})q(\sigma_{1:T}) \,.$$

From the Kullback-Leibler divergence between this distribution and the original posterior distribution we obtain a tractable lower bound on the log-likelihood $\log p(v_{1:T}|\Theta)$, given by

$$\mathcal{B}(q,\Theta) = H_{q(h_{1:M}^{1:S})} + \langle H_{q(z_{1:T}|\sigma_{1:T})} \rangle_{q(\sigma_{1:T})} + H_{q(\sigma_{1:T})}$$
$$+ \langle \log p(v_{1:T}|h_{1:M}^{1:S}, z_{1:T}, \sigma_{1:T}, \Theta) \rangle_{q(h_{1:M}^{1:S})q(z_{1:T}|\sigma_{1:T})q(\sigma_{1:T})}$$
$$+ \langle \log p(z_{1:T}|\sigma_{1:T},\Theta) \rangle_{q(z_{1:T}|\sigma_{1:T})q(\sigma_{1:T})} + \langle \log p(\sigma_{1:T}|\Theta) \rangle_{q(\sigma_{1:T})} + \langle \log p(h_{1:M}^{1:S}|\Theta) \rangle_{q(h_{1:M}^{1:S})} \,,$$

where $\langle \cdot \rangle_q$ denotes expectation with respect to $q$, and $H_q$ denotes the entropy of $q$. We then use a variational EM algorithm in which $\mathcal{B}(q,\Theta)$ is iteratively maximized with respect to $q$ and the model parameters $\Theta$ until convergence[7].

Maximization with respect to $q$ leads to the following updates

$$q(h_{1:M}^{1:S}) \propto p(h_{1:M}^{1:S})e^{\left\langle \log p(v_{1:T}|h_{1:M}^{1:S}, z_{1:T}, \sigma_{1:T}) \right\rangle_{q(z_{1:T}|\sigma_{1:T})q(\sigma_{1:T})}} \,, \tag{1}$$

$$q(\sigma_{1:T}) \propto p(\sigma_{1:T})e^{H_{q(z_{1:T}|\sigma_{1:T})}}e^{\left\langle \log p(v_{1:T}, z_{1:T}|h_{1:M}^{1:S}, \sigma_{1:T}) \right\rangle_{q(h_{1:M}^{1:S})q(z_{1:T}|\sigma_{1:T})}} \,, \tag{2}$$

$$q(z_{1:T}|\sigma_{1:T}) \propto p(z_{1:T}|\sigma_{1:T})e^{\left\langle \log p(v_{1:T}|h_{1:M}^{1:S}, z_{1:T}, \sigma_{1:T}) \right\rangle_{q(h_{1:M}^{1:S})}} \,. \tag{3}$$

Before describing how to perform inference on these distributions, we observe that all quantities required for learning $\Theta$, sampling a segmentation, and updating $q(h_{1:M}^{1:S})$ can be formulated such that only partial inference on $q(\sigma_{1:T})$ and $q(z_{1:T}|\sigma_{1:T})$ is required. For example, we can write

$$\left\langle \log p(v_{1:T}|h_{1:M}^{1:S}, z_{1:T}, \sigma_{1:T}) \right\rangle_{q(z_{1:T}, \sigma_{1:T})} = \sum_{t,i,k} \gamma_t^{i,k,1} \sum_{\tau,m} \xi_\tau^{i,k,t,m} \log p(v_\tau|h_m^i, z_\tau = m, \sigma_t^{i,k,1}), \tag{4}$$

with $\gamma_t^{i,k,1} = q(\sigma_t^{i,k,1})$, $\xi_\tau^{i,k,t,m} = q(z_\tau = m|\sigma_t^{i,k,1})$, $\sigma_t^{i,k,1} = \{s_t = i, d_t = k, c_t = 1\}$. Thus, only posteriors for which the count variables take value 1 are required[8].

**Inference on $q(h_{1:M}^{1:S})$.** We first notice that, by using (4) in (1), we obtain $q(h_{1:M}^{1:S}) = \prod_i q(h_{1:M}^i)$. We then observe that we can rewrite the update for $q(h_{1:M}^i)$ as proportional to the joint distribution of the following linear gaussian state-space model (LGSSM)

$$h_m^i = F^i h_{m-1}^i + \eta_m^h, \, \eta_m^h \sim \mathcal{N}(0, \Sigma_H^i), \, h_1^i \sim \mathcal{N}(\mu^i, \Sigma^i), \, \hat{v}_m^i = G^i h_m^i + \eta_m^v, \, \eta_m^v \sim \mathcal{N}(0, \hat{\Sigma}_{V,m}^i),$$

where

$$\hat{v}_m^i \equiv 1/a_m^i \sum_{t,k} \gamma_t^{i,k,1} \sum_{\tau=t-k+1}^{t} \xi_\tau^{i,k,t,m} v_\tau, \quad \hat{\Sigma}_{V,m}^i \equiv 1/a_m^i \Sigma_V^i, \quad a_m^i \equiv \sum_{t,k} \gamma_t^{i,k,1} \sum_{\tau=t-k+1}^{t} \xi_\tau^{i,k,t,m}.$$

Therefore, inference on $q(h_{1:M}^{1:S})$ can be accomplished with LGSSM smoothing routines [7].

**Inference on** $q(\sigma_{1:T})$**.** By substituting update (3) (including the normalization constant) into update (2), we obtain $q(\sigma_{1:T}) \propto q(v_{1:T}|\sigma_{1:T})p(\sigma_{1:T})$. This update has the form of the joint distribution of an ERDM using separate duration and count variables [4]. Therefore, we can employ similar forward-backward recursions. More specifically $\gamma_t^{i,k,1} = q(\sigma_t^{i,k,1}) = q(v_{t+1:T}|s_t = j, c_t = 1)q(\sigma_t^{i,k,1}, v_{1:t})/q(v_{1:T}) = \beta_t^{i,1}\alpha_t^{i,k,1}/q(v_{1:T})$, where

$$\alpha_t^{i,k,1} = q(v_{t-k+1:t}|\sigma_t^{i,k,1}) \sum_{jl} \underbrace{p(\sigma_t^{i,k,1}|\sigma_{t-k}^{j,l,1})}_{\rho_k \pi_{i,j}} \alpha_{t-k}^{j,l,1}, \quad \beta_t^{j,1} = \sum_{i,k} q(v_{t+1:t+k}|\sigma_{t+k}^{i,k,1})\pi_{i,j}\beta_{t+k}^{i,1}\rho_k.$$

Since we have imposed the constraints $c_0 = 1, c_T = 1$, we need to replace terms such as $p(d_t = k, c_t = 1|c_{t-k} = 1) = \rho_k$ with $p(d_t = k, c_t = 1|c_{t-k} = 1, c_0 = 1, c_T = 1)$. The constraint $c_T = 1$ implies $q(v_{1:T}) = \sum_{j,l} \alpha_T^{j,l,1}$.

Required terms such as $q(v_{t-k+1:t}|\sigma_t^{i,k,1})$ can be computed as likelihood terms when performing inference on $q(z_{t-k+1:t}|\sigma_t^{i,k,1})$.

**Inference on** $q(z_{1:T}|\sigma_{1:T})$**.** The form of update (3) implies that inference on distributions of the type $q(z_{t-k+1:t}|\sigma_t^{i,k,1})$ can be accomplished with forward-backward routines similar to the ones used in hidden Markov models (HMMs).

**Sampling a segmentation.** A segmentation can be sampled by using the factorization $q(\sigma_{1:T}|v_{1:T}) = q(\sigma_T|v_{1:T}) \prod_{t=1}^{T-1} q(\sigma_t|\sigma_{t+1}, v_{1:T})$, with

$$q(\sigma_t|\sigma_{t+1}, v_{1:T}) = \frac{p(\sigma_{t+1}|\sigma_t)q(v_{t+1}|\sigma_{t+1})\alpha_t^{\sigma_t}\beta_{t+1}^{\sigma_{t+1}}}{\alpha_{t+1}^{\sigma_{t+1}}\beta_{t+1}^{\sigma_{t+1}}}.$$

Suppose that, at time $t$, $c_t = 1$ and we have sampled dynamics type $s_t = i$ and duration $d_t = k$. Then, $\sigma_{t-k+1:t-1}$ and $c_{t-k}$ are determined by the model assumptions[9], so that we effectively need to sample $s_{t-k}, d_{t-k}$ from the distribution $q(s_{t-k}, d_{t-k}, c_{t-k} = 1|\sigma_{t-k+1}, v_{1:T})$, which is given by

$$\rho_k \pi_{i,:} q(v_{t-k+1:t}|\sigma_t)\alpha_{t-k}^{:,:,1}/\alpha_t^{i,k,1},$$

since $q(v_{t-k+2:t}|\sigma_{t-k+2:t})\alpha_{t-k+1}^{i,k,k} = \alpha_t^{i,k,1}$.

## 3.2   Maximum a Posteriori (MAP) Method

Instead of approximating the posterior distribution of all hidden variables, we can approximate only $p(h_{1:M}^{1:S}|v_{1:T})$ with a deterministic distribution, by using the variational method described above in which $q(h_m^i)$ is a Dirac delta around its mean. Notice that this is equivalent to compute the most likely set of hidden trajectories and parameters by maximizing the joint distribution $p(v_{1:T}, h_{1:M}^{1:S}|\Theta)$ with respect to $h_{1:M}^{1:S}$ and $\Theta$ using an EM algorithm.

## 3.3   Gibbs Sampling Method

In our stochastic EM approach with Gibbs sampling, the expectation of the complete-data log-likelihood $\mathcal{L}(\Theta)$ is approximated by $\mathcal{L}(\Theta) \approx \sum_{n=1}^{N} \log p(v_{1:T}, \hat{z}_{1:T}^n, \hat{\sigma}_{1:T}^n, \hat{h}_{1:M}^{1:S,n}|\Theta)$, where $\hat{z}_{1:T}^n, \hat{\sigma}_{1:T}^n, \hat{h}_{1:M}^{1:S,n}$ are samples drawn from $p(h_{1:M}^{1:S}, z_{1:T}, \sigma_{1:T}|v_{1:T})$. Such samples can be obtained by iterative drawing from the tractable conditionals

$$p(z_{1:T}, \sigma_{1:T}|h_{1:M}^{1:S}, v_{1:T}) \text{ and } p(h_{1:M}^{1:S}|z_{1:T}, \sigma_{1:T}, v_{1:T}).$$

| | Time-series 1 | Time-series 2 | Time-series 3 | Time-series 4 | Time-series 5 |
|---|---|---|---|---|---|
| Correct Seg. | 1 24 42 66 89 | 1 23 46 63 | 1 23 40 63 | 1 22 47 68 | 1 24 42 65 88 105 |
| Variational Approx. | 1 17 39 62 82 | 1 23 46 64 | 1 21 39 62 | 1 23 47 68 | 1 18 42 65 82 100 |
| | 1 17 39 63 82 | 1 18 42 63 | 1 22 38 62 | 1 22 47 66 | 1 18 42 65 82 99 |
| | 1 17 38 62 81 | 1 18 42 63 | 1 17 38 60 87 | 1 9 23 31 60 66 | 1 6 12 42 58 76 83 100 |
| | 1 14 39 63 79 | 1 22 45 63 | 1 23 38 62 | 1 9 23 31 46 67 | 1 11 18 42 60 85 102 |
| MAP Approx. | 1 20 40 64 85 | 1 23 46 62 | 1 21 40 64 | 1 22 47 69 | 1 18 40 55 65 82 97 |
| | 1 19 40 64 84 | 1 23 46 62 | 1 21 40 63 | 1 22 47 68 | 1 18 42 64 82 98 |
| | 1 17 39 63 82 | 1 23 46 62 | 1 22 40 65 | 1 22 47 68 | 1 18 42 65 82 96 |
| | 1 20 40 64 85 | 1 23 46 63 | 1 22 40 63 | 1 15 20 45 67 | 1 11 19 40 60 85 102 |
| Gibbs Sampling Approx. | 1 17 39 63 82 | 1 18 36 56 | 1 23 38 62 | 1 14 24 38 63 | 1 16 44 71 97 |
| | 1 22 41 64 88 | 1 20 42 60 | 1 16 35 61 82 | 1 14 24 38 63 | 1 16 40 63 80 102 |
| | 1 17 40 65 81 | 1 9 23 47 63 71 | 1 17 40 64 84 | 1 9 22 32 47 68 | 1 22 44 63 89 104 |
| | 1 17 40 64 82 | 1 21 47 62 | 1 17 37 60 86 | 1 9 23 31 52 74 | 1 7 13 21 31 58 71 101 114 |

Table 2: Segmentations given by the variational, MAP and Gibbs sampling methods on 5 artificial time-series.

In order to sample from $p(z_{1:T}, \sigma_{1:T}|h_{1:M}^{1:S}, v_{1:T})$, we can first sample a segmentation from $p(\sigma_{1:T}|h_{1:M}^{1:S}, v_{1:T})$ employing the method described above (with $q(\cdot)$ replaced by $p(\cdot|h_{1:M}^{1:S}, v_{1:T})$), and then use a HMM forward-filtering backward-sampling method for sampling from $p(z_{1:T}|\sigma_{1:T}, h_{1:M}^{1:S}, v_{1:T})$. Finally, sampling from $p(h_{1:M}^{1:S}|z_{1:T}, \sigma_{1:T}, v_{1:T})$ may be carried out using the forward-filtering backward-sampling procedure described in [8].

### 3.4 Comparison of the Approximation Methods

In this section, we compare the performance of the approximation methods presented above on 5 artificially generated time-series. Each time-series (with V=2 or V=3) contains repeated occurrences of actions arising from the noisy transformation of up to three hidden trajectories with time-warping.

In the second row of Table 2, we give the correct segmentation for each time-series. Each number indicates the time-step at which a new action starts, whilst the colors indicate the types of dynamics underlying the actions. In the rows below, we give the segmentations obtained by each approximation method with 4 different initial random conditions (with minimum and maximum action duration between 5 and 30).

From the results, we can deduce that Gibbs sampling performs considerably worse than the deterministic approaches. Between the variational and MAP methods, the latter is preferable and gives a good solution in most cases. The poor performance of Gibbs sampling can be explained by the fact that this method cannot deal well with high correlation between $h_{1:M}^{1:S}$ and $\sigma_{1:T}, z_{1:T}$. The continuous hidden variables are sampled given a single set of segmentation and time-warping variables (unlike update (1) in which we average over segmentation and time-warping variables), which may result in poor mixing. The inferior performance of the variational method in comparison to the MAP method would seem to suggest that the posterior covariances of the continuous hidden variables cannot accurately be estimated.

## 4 Table Tennis Recordings using a Robot Arm

In this section, we show how the proposed model performs in segmenting time-series obtained from table tennis recordings using a robot arm moved by a human. The generic goal is to extract movement templates to be used for robot imitation learning [2, 9]. Here, kinesthetic teach-in can be advantageous in order to avoid the correspondence problem.

We used the Barrett WAM robot arm shown in Figure 2 as a haptic input device for recording and replaying movements. We recorded a game of table tennis where a human moved the robot arm making the typical moves occurring in this specific setup. These naturally include forehands, going into an awaiting posture for a forehand, backhands, and going into an awaiting posture for a backhand. They also include smashes, however, due to the inertia of the robot, they are hard to perform and only occur using the forehand.

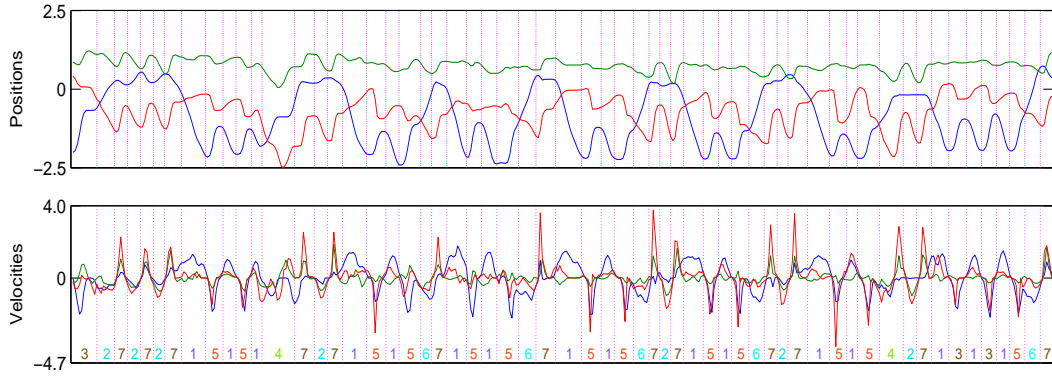

Figure 3: This figure shows the first three degrees of freedom (known as flexion-extension, adduction-abduction and humerus rotation) of a robot arm when used by a human as a haptic input device playing table tennis. The upper graph shows the joint positions while the lower one shows the joint velocities. The dashed vertical lines indicate the obtained action boundaries and the numbers the underlying movement templates. This sequence includes moves to the right awaiting posture (1), moves to the left awaiting posture (2), forehands (3, 5), two incomplete moves towards the awaiting posture merged with a backhand (4), moves to the left awaiting posture with humerus rotation (6) and backhands (7).

The recorded time-series contains the joint positions and velocities of all seven degrees of freedom (DoF) of an anthropomorphic arm. However, only the shoulder and upper arm DoF, which are the most significant in such movements, were considered for the analysis. The 1.5 minutes long recording was sub-sampled at 5 samples per seconds. The minimum and maximum durations $d_{\min}$ and $d_{\max}$ were set to 4 and 15 respectively, as prior knowledge about table tennis would suggest that basic-action durations are within this range. We also imposed the constraint that nearly complete movements are observed ($\iota = 2, \epsilon = M - 1$). The length of the hidden dynamics $M$ was set to $d_{\max}$, the variable $w_{\max}$ was set to[10] 4, and the number of movement templates $S$ was set to 8, as this should be a reasonable upper bound on the number of different underlying movements. Given the results obtained in the previous section, we used the MAP approximation method.

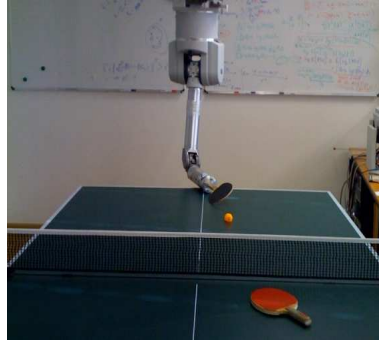

Figure 2: The Barrett WAM used for recording the table tennis sequences. During the experiment the robot is in a gravity compensation and sequences can be replayed on the real system.

We assumed no prior knowledge on the dynamics of the hidden trajectories. However, in a real application of the model we could simplify the problem by incorporating knowledge about previously identified movement templates.

As shown in Figure 3, the model segments the time-series into 59 basic movements of forehands (numbers 3, 5), backhands (7), and going into a right (1) and left (2, 6) awaiting posture. In some cases, a more fluid game results in incomplete moves towards an awaiting posture and hence into a composite movement that can no longer be segmented (4). Also, there appear to be two types of moving back to the left awaiting posture: one which needs untwisting of the humerus rotation degree of freedom (6), and another which purely employs shoulder degrees of freedom (2).

The action boundaries estimated by the model are in strong agreement with manual visual segmentation, with the exception of movements 4 that should be segmented into two separate movements. At the web-page http://silviac.yolasite.com we provide a visual interpretation of the segmentation from which the model accuracy can be appreciated.

## 5  Conclusions

In this paper we have introduced a probabilistic model for detecting repeated occurrences of basic movements in time-series data. This model may potentially be applicable in domains such as robotics, sports sciences, physical therapy, virtual reality, artificial movie generation, computer games, etc., for automatic extraction of the movement templates contained in a recording. We have presented an evaluation on table tennis movements that we have recorded using a robot arm as haptic input device, showing that the model is able to accurately segment the time-series into basic movements that could be used for robot imitation learning.

## Appendix

**Constraints on $z_{1:T}$**

Consider an action starting at time 1 and finishing at time $t$ with the constraints $z_1 \in \{1, \ldots, \iota\}$ and $z_t \in \{\epsilon, \ldots, M\}$. Suppose that $z_\tau = m$ for $\tau \in \{1, \ldots, t-1\}$. Then it must be

1. $m \in \{\max[\tau, \epsilon - (t-\tau)w_{\max}], \ldots, \min[\iota + (\tau-1)w_{\max}, M - (t-\tau)]\}$.
2. $z_{\tau+1} \in \{\max[m+1, \epsilon - (t-\tau-1)w_{\max}], \ldots, \min[m + w_{\max}, M - (t-\tau-1)]\}$.

Therefore, we need to modify the original priors $\tilde{\psi}, \psi$ with time-dependent priors with zero values outside the appropriate range.

## Footnotes

[1]For the sake of notational simplicity, we describe the model for the case of a single observed time-series and hidden trajectories of the same length $M$.

[2]We assume $c_0 = 1$, $c_T = 1$. For $t = 1$, $p(s_1) = \tilde{\pi}_{s_1}, p(d_1) = \rho_{d_1}, p(c_1|d_1) = \delta(c_1 = d_1)$.

[3] In the experiments we added the additional constraint that nearly complete movements are observed, that is $z_{t-d_t+c_t} \in \{1, \ldots, \iota\}, z_{t+c_t-1} \in \{\epsilon, \ldots, M\}$ (see the Appendix for more details).

[4] With $\pi_{:, s_{t-1}}$ we indicate the vector of transition probabilities from dynamics type $s_{t-1}$ to any dynamics.

[5] Due to space limitations, we describe only how to sample a segmentation, which is required in the Gibbs sampling method.

[6]Conditioning on $v_{1:T}$ in $q$ is omitted for notational simplicity.

[7]Maximization with respect to $\Theta$ is omitted due to space limitations.

[8]This is common to ERDMs using separate duration and count variables [4].

[9]The values of $c_{1:T-1}$ are automatically determined if $c_T$ and $d_{1:T}$ are given.

[10]This is the smallest value that ensures that complete actions can be observed.

## References

[1] R. Boulic, B. Ulicny, and D. Thalmann. Versatile walk engine. *Journal of Game Development*, 1(1):29–52, 2004.

[2] S. Calinon, F. Guenter, and A. Billard. On learning, representing and generalizing a task in a humanoid robot. *IEEE Transactions on Systems, Man and Cybernetics, Part B*, 37(2):286–298, 2007.

[3] G. Celeux and J. Diebolt. The SEM algorithm: A probabilistic teacher algorithm derived from the EM algorithm for the mixture problem. *Computational Statistics Quarterly*, 2:73–82, 1985.

[4] S. Chiappa. Hidden Markov switching models with explicit regime-duration distribution. Under submission.

[5] S. Chiappa, J. Kober, and J. Peters. Using Bayesian dynamical systems for motion template libraries. In *Advances in NIPS 21*, pages 297–304, 2009.

[6] A. Coates, P. Abbeel, and A. Y. Ng. Learning for control from multiple demonstrations. In *Proceedings of ICML*, pages 144–151, 2008.

[7] J. Durbin and S. J. Koopman. *Time Series Analysis by State Space Methods*. Oxford Univ. Press, 2001.

[8] S. Frühwirth-Schnatter. Data augmentation and dynamic linear models. *Journal of Time-Series Analysis*, 15:183–202, 1994.

[9] A. Ijspeert, J. Nakanishi, and S. Schaal. Learning attractor landscapes for learning motor primitives. In *Advances in NIPS 15*, pages 1547–1554, 2003.

[10] U. Kersting, P. McAlpine, B. Rosenhahn, H. Seidel, and R. Klette. Marker-less human motion tracking opportunities for field testing in sports. *Journal of Biomechanics*, 39:S191–S191, 2006.

[11] J. Listgarten, R. M. Neal, S. T. Roweis, and A. Emili. Multiple alignment of continuous time series. In *Advances in NIPS 17*, pages 817–824, 2005.

[12] J. Listgarten, R. M. Neal, S. T. Roweis, R. Puckrin, and S. Cutler. Bayesian detection of infrequent differences in sets of time series with shared structure. In *Advances in NIPS 19*, pages 905–912, 2007.

[13] R. McDonnell, S. J:org, J. K. Hodgins, F. N. Newell, and C. O'Sullivan. Evaluating the effect of motion and body shape on the perceived sex of virtual characters. *ACM Transactions on Applied Perception*, 5(4), 2009.

[14] W. Pan and L. Torresani. Unsupervised hierarchical modeling of locomotion styles. In *Proceedings of ICML*, 2009.

[15] M. Peinado, D. Maupu, D. Raunhardt, D. Meziat, D. Thalmann, and R. Boulic. Full-body avatar control with environment awareness. *IEEE Computer Graphics and Applications*, 29(3), 2009.

[16] W. Takano, K. Yamane, and Y. Nakamura. Capture database through symbolization, recognition and generation of motion patterns. In *Proceedings of ICRA*, pages 3092–3097, 2007.

[17] B. Williams, M. Toussaint, and A. Storkey. Modelling motion primitives and their timing in biologically executed movements. In *Advances in NIPS 20*, pages 1609–1616, 2008.

[18] K. Yamane and J. K. Hodgins. Simultaneous tracking and balancing of humanoid robots for imitating human motion capture data. In *Proceedings of IROS*, pages 2510–2517, 2009.

